# Truncation-free Stochastic Variational Inference for Bayesian Nonparametric Models

**Chong Wang**[*]
Machine Learning Department
Carnegie Mellon University
chongw@cs.cmu.edu

**David M. Blei**
Computer Science Department
Princeton Univeristy
blei@cs.princeton.edu

## Abstract

We present a truncation-free stochastic variational inference algorithm for Bayesian nonparametric models. While traditional variational inference algorithms require truncations for the model or the variational distribution, our method adapts model complexity on the fly. We studied our method with Dirichlet process mixture models and hierarchical Dirichlet process topic models on two large data sets. Our method performs better than previous stochastic variational inference algorithms.

## 1 Introduction

Bayesian nonparametric (BNP) models [1] have emerged as an important tool for building probability models with flexible latent structure and complexity. BNP models use posterior inference to adapt the model complexity to the data. For example, as more data are observed, Dirichlet process (DP) mixture models [2] can create new mixture components and hierarchical Dirichlet process (HDP) topic models [3] can create new topics.

In general, posterior inference in BNP models is intractable and we must approximate the posterior. The most widely-used approaches are Markov chain Monte Carlo (MCMC) [4] and variational inference [5]. For BNP models, the advantage of MCMC is that it directly operates in the unbounded latent space; whether to increase model complexity (such as adding a new mixture component) naturally folds in to the sampling steps [6, 3]. However MCMC does not easily scale—it requires storing many configurations of hidden variables, each one on the order of the number of data points. For scalable MCMC one typically needs parallel hardware, and even then the computational complexity scales linearly with the data, which is not fast enough for massive data [7, 8, 9].

The alternative is variational inference, which finds the member of a simplified family of distributions to approximate the true posterior [5, 10]. This is generally faster than MCMC, and recent innovations let us use stochastic optimization to approximate posteriors with massive and streaming data [11, 12, 13]. Unlike MCMC, however, variational inference algorithms for BNP models do not operate in an unbounded latent space. Rather, they truncate the model or the variational distribution to a maximum model complexity [13, 14, 15, 16, 17, 18].[1] This is particularly limiting in the stochastic approach, where we might hope for a Bayesian nonparametric posterior seamlessly adapting its model complexity to an endless stream of data.

In this paper, we develop a truncation-free stochastic variational inference algorithm for BNP models. This lets us more easily apply Bayesian nonparametric data analysis to massive and streaming data.

---

[*]Work was done when the author was with Princeton University.

[1]In [17, 18], split-merge techniques were used to grow/shrink truncations. However, split-merge operations are model-specific and difficult to design. It is also unknown how to apply these to the stochastic variational inference setting we consider.

In particular, we present a new general inference algorithm, *locally collapsed variational inference*. When applied to BNP models, it does not require truncations and gives a principled mechanism for increasing model complexity on the fly. We demonstrate our algorithm on DP mixture models and HDP topic models with two large data sets, showing improved performance over truncated algorithms.

## 2 Truncation-free stochastic variational inference for BNP models

Although our goal is to develop an efficient *stochastic variational inference* algorithm for BNP models, it is more succinct to describe our algorithm for a wider class of hierarchical Bayesian models [19]. We will show how we apply our algorithm for BNP models in §2.3.

We consider the general class of hierarchical Bayesian models shown in Figure 1. Let the *global* hidden variables be $\beta$ with prior $p(\beta \mid \eta)$ ($\eta$ is the hyperparameter) and *local* variables for each data sample be $z_i$ (hidden) and $x_i$ (observed) for $i = 1, \ldots, n$. The joint distribution of all variables (hidden and observed) factorizes as,

$$p(\beta, z_{1:n}, x_{1:n} \mid \eta) = p(\beta \mid \eta) \prod_{i=1}^{n} p(x_i, z_i \mid \beta) = p(\beta \mid \eta) \prod_{i=1}^{n} p(x_i \mid z_i, \beta) p(z_i \mid \beta). \quad (1)$$

The idea behind the nomenclature is that the local variables are conditionally independent of each other given the global variables. For convenience, we assume global variables $\beta$ are continuous and local variables $z_i$ are discrete. (This assumption is not necessary.) A large range of models can be represented using this form, e.g., mixture models [20, 21], mixed-membership models [3, 22], latent factor models [23, 24] and tree-based hierarchical models [25].

As an example, consider a DP mixture model for document clustering. Each document is modeled as a bag of words drawn from a distribution over the vocabulary. The mixture components are the distributions over the vocabulary $\theta$ and the mixture proportions $\pi$ are represented with a stick-breaking process [26]. The global variables $\beta \triangleq (\pi, \theta)$ contain the proportions and components, and the local variables $z_i$ are the mixture assignments for each document $x_i$. The generative process is:

1. Draw mixture component $\theta_k$ and sticks $\pi_k$ for $k = 1, 2, \cdots$,

$$\theta_k \sim \text{Dirichlet}(\eta), \ \ \pi_k = \bar{\pi}_k \prod_{\ell=1}^{k-1} (1 - \bar{\pi}_\ell), \ \ \bar{\pi}_k \sim \text{Beta}(1, a).$$

2. For each document $x_i$,
   (a) Draw mixture assignment $z_i \sim \text{Mult}(\pi)$.
   (b) For each word $x_{ij}$, draw the word $x_{ij} \sim \text{Mult}(\theta_{z_i})$.

We now return to the general model in Eq. 1. In inference, we are interested in the posterior of the hidden variables $\beta$ and $z_{1:n}$ given the observed data $x_{1:n}$, i.e., $p(\beta, z_{1:n} \mid x_{1:n}, \eta)$. For many models, this posterior is intractable. We will approximate it using mean-field variational inference.

### 2.1 Variational inference

In variational inference we try to find a distribution in a simple family that is close to the true posterior. We describe the mean-field approach, the simplest variational inference algorithm [5]. It assumes the fully factorized family of distributions over the hidden variables,

$$q(\beta, z_{1:n}) = q(\beta) \prod_{i=1}^{n} q(z_i). \quad (2)$$

We call $q(\beta)$ the global variational distribution and $q(z_i)$ the local variational distribution. We want to minimize the KL-divergence between this variational distribution and the true posterior. Under the standard variational theory [5], this is equivalent to maximizing a lower bound of the log marginal likelihood of the observed data $x_{1:n}$. We obtain this bound with Jensen's inequality,

$$\log p(x_{1:n} \mid \eta) = \log \int \sum_{z_{1:n}} p(x_{1:n}, z_{1:n}, \beta \mid \eta) \mathrm{d}\beta$$
$$\geq \mathbb{E}_q \left[ \log p(\beta) - \log q(\beta) + \sum_{i=1}^{n} \log p(x_i, z_i \mid \beta) - \log q(z_i) \right] \triangleq \mathcal{L}(q). \quad (3)$$

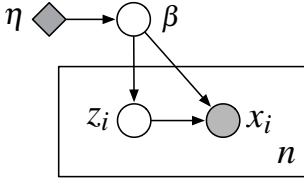

Figure 1: Graphical model for hierarchical Bayesian models with global hidden variables $\beta$, local hidden and observed variables $z_i$ and $x_i$, $i = 1, \ldots, n$. Hyperparameter $\eta$ is fixed, not a random variable.

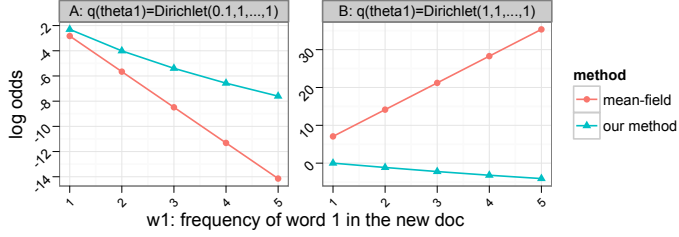

Figure 2: Results on assigning document $d = \{w_1, 0, \ldots, 0\}$ to $q(\theta_1)$ (case A and B, shown in the figure above) or $q(\theta_2) = \mathrm{Dirichlet}(0.1, 0.1, \ldots, 0.1)$. The $y$ axis is the log-odds of $q(z = 1)$ to $q(z = 2)$—if it is larger than 0, it is more likely to be assigned to component 1. The mean-field approach underestimates the uncertainty around $\theta_2$, assigning $d$ incorrectly for case B. The locally collapsed approach does it correctly in both cases.

---

**Algorithm 1** Mean-field variational inference.

1: Initialize $q(\beta)$.
2: **for** iter $= 1$ to $M$ **do**
3:     **for** $i = 1$ to $n$ **do**
4:         Set local variational distribution $q(z_i) \propto \exp\left\{\mathbb{E}_{q(\beta)}\left[\log p(x_i, z_i \,|\, \beta)\right]\right\}$.
5:     **end for**
6:     Set global variational distribution $q(\beta) \propto \exp\left\{\mathbb{E}_{q(z_{1:n})}\left[\log p(x_{1:n}, z_{1:n}, \beta)\right]\right\}$.
7: **end for**
8: **return** $q(\beta)$.

**Algorithm 2** Locally collapsed variational inference.

1: Initialize $q(\beta)$.
2: **for** iter $= 1$ to $M$ **do**
3:     **for** $i = 1$ to $n$ **do**
4:         Set local distribution $q(z_i) \propto \mathbb{E}_{q(\beta)}\left[p(x_i, z_i \,|\, \beta)\right]$.
5:         Sample from $q(z_i)$ to obtain its empirical $\hat{q}(z_i)$.
6:     **end for**
7:     Set global variational distribution $q(\beta) \propto \exp\left\{\mathbb{E}_{\hat{q}(z_{1:n})}\left[\log p(x_{1:n}, z_{1:n}, \beta)\right]\right\}$.
8: **end for**
9: **return** $q(\beta)$.

---

Maximizing $\mathcal{L}(q)$ w.r.t. $q(\beta, z_{1:n})$ defined in Eq. 2 (with the optimal conditions given in [27]) gives

$$q(\beta) \propto \exp\left\{\mathbb{E}_{q(z_{1:n})}\left[\log p(x_{1:n}, z_{1:n}, \beta \,|\, \eta)\right]\right\} \qquad (4)$$

$$q(z_i) \propto \exp\left\{\mathbb{E}_{q(\beta)}\left[\log p(x_i, z_i \,|\, \beta)\right]\right\}. \qquad (5)$$

Typically these equations are used in a coordinate ascent algorithm, iteratively optimizing each factor while holding the others fixed (see Algorithm 1). The factorization into global and local variables ensures that the local updates only depend on the global factors, which facilitates speed-ups like parallel [28] and stochastic variational inference [11, 12, 13, 29].

In BNP models, however, the value of $z_i$ is potentially unbounded (e.g., the mixture assignment in a DP mixture). Thus we need to truncate the variational distribution [13, 14]. Truncation is necessary in variational inference because of the mathematical structure of BNP models. Moreover, it is difficult to grow the truncation in mean-field variational inference even in an ad-hoc way because it tends to underestimate posterior variance [30, 31]. In contrast, its mathematical structure and that it gets the variance right in the conditional distribution allow Gibbs sampling for BNP models to effectively explore the unbounded latent space [6].

## 2.2 Locally collapsed variational inference

We now describe *locally collapsed variational inference*, which mitigates the problem of underestimating posterior variance in mean-field variational inference. Further, when applied to BNP models, it is truncation-free—it gives a good mechanism to increase truncation on the fly.

Algorithm 2 outlines the approach. The difference between traditional mean-field variational inference and our algorithm lies in the update of the local distribution $q(z_i)$. In our algorithm, it is

$$q(z_i) \propto \mathbb{E}_{q(\beta)}\left[p(x_i, z_i \,|\, \beta)\right], \qquad (6)$$

as opposed to the mean-field update in Eq. 5. Because we collapse out the global variational distribution $q(\beta)$ locally, we call this method *locally collapsed variational inference*. Note the two algorithms are similar when $q(\beta)$ has low variance. However, when the uncertainty modeled in $q(\beta)$ is high, these two approaches lead to different approximations of the posterior.

In our implementation, we use a collapsed Gibbs sampler to sample from Equation 6. This is a local Gibbs sampling step and thus is very fast. Further, this is where our algorithm does not require truncation because Gibbs samplers for BNP models can operate in an unbounded space [6, 3].

Now we update $q(\beta)$. Suppose we have a set of samples from $q(z_i)$ to construct its empirical distribution $\hat{q}(z_i)$. Plugging this into Eq. 3 gives the solution to $q(\beta)$,

$$q(\beta) \propto \exp\left\{\mathbb{E}_{\hat{q}(z_{1:n})}\left[\log p(x_{1:n}, z_{1:n}, \beta \mid \eta)\right]\right\}, \tag{7}$$

which has the same form as in Eq. 4 for the mean-field approach. This finishes Algorithm 2.

To give an intuitive comparison of locally collapsed (Algorithm 2) and mean-field (Algorithm 1) variational inference, we consider a toy document clustering problem with vocabulary size $V = 10$. We use a two-component Bayesian mixture model with fixed and equal prior proportions $\pi_1 = \pi_2 = 1/2$. Suppose at some stage, component 1 has some documents assignments while component 2 has not yet and we have obtained the (approximate) posterior for the two component parameters $\theta_1$ and $\theta_2$ as $q(\theta_1)$ and $q(\theta_2)$. For $\theta_1$, we consider two cases, A) $q(\theta_1) = \text{Dirichlet}(0.1, 1, \ldots, 1)$; B) $q(\theta_1) = \text{Dirichlet}(1, 1, \ldots, 1)$. For $\theta_2$, we only consider $q(\theta_2) = \text{Dirichlet}(0.1, 0.1, \ldots, 0.1)$. In both cases, $q(\theta_1)$ has relatively low variance while $q(\theta_2)$ has high variance. The difference is that the $q(\theta_1)$ in case A has a lower probability on word 1 than that in case B. Now we have a new document $d = \{w_1, 0, \ldots, 0\}$, where word 1 is the only word and its frequency is $w_1$. In both cases, document $d$ is more likely be assigned to component 2 when $w_1$ becomes larger. Figure 2 shows the difference between mean-field and locally collapsed variational inference. In case A, the mean-field approach does it correctly, since word 1 already has a very low probability in $\theta_1$. But in case B, it ignores the uncertainty around $\theta_2$, resulting in incorrect clustering. Our approach does it correctly in both cases.

What justifies this approach? Alas, as for some other adaptations of variational inference, we do not yet have an airtight justification [32, 33, 34]. We are not optimizing $q(z_i)$ and so the corresponding lower bound must be looser than the optimized lower bound from the mean-field approach if the issue of local modes is excluded. However, our experiments show that we find a better predictive distribution than mean-field inference. One possible explanation is outlined in S.1 (section 1 of the supplement), where we show that our algorithm can be understood as an approximate Expectation Propagation (EP) algorithm [35].

**Related algorithms.** Our algorithm is closely related to collapsed variational inference (CVI) [15, 16, 36, 32, 33]. CVI applies variational inference to the marginalized model, integrating out the global hidden variable $\beta$. This gives better estimates of posterior variance. In CVI, however, the optimization for each local variable $z_i$ depends on all other local variables, and this makes it difficult to apply it at large scale. Our algorithm is akin to applying CVI for the intermediate model that treats $q(\beta)$ as a prior and considers a single data point $x_i$ with its hidden structure $z_i$. This lets us develop stochastic algorithms that can be fit to massive data sets (as we show below).

Our algorithm is also related to the recently proposed a hybrid approach of using Gibbs sampling inside stochastic variational inference to take advantage of the sparsity in text documents in topic modeling [37]. Their approach still uses the mean-field update as in Eq. 5, where all local hidden topic variables (for a document) are grouped together and the optimal $q(z_i)$ is approximated by a Gibbs sampler. With some adaptations, their fast sparse update idea can be used inside our algorithm.

**Stochastic locally collapsed variational inference.** We now extend our algorithm to stochastic variational inference, allowing us to fit approximate posteriors to massive data sets. To do this, we assume the model in Figure 1 is in the exponential family and satisfies the *conditional conjugacy* [11, 13, 29]—the global distribution $p(\beta \mid \eta)$ is the conjugate prior for the local distribution $p(x_i, z_i \mid \beta)$,

$$p(\beta \mid \eta) = h(\beta) \exp\left\{\eta^\top t(\beta) - a(\eta)\right\}, \tag{8}$$

$$p(x_i, z_i \mid \beta) = h(x_i, z_i) \exp\left\{\beta^\top t(x_i, z_i) - a(\beta)\right\}, \tag{9}$$

where we overload the notation for base measures $h(\cdot)$, sufficient statistics $t(\cdot)$, and log normalizers $a(\cdot)$. (These will often be different for the two families.) Due to the conjugacy, the term $t(\beta)$ has the form $t(\beta) = [\beta; -a(\beta)]$. Also assume the global variational distribution $q(\beta \mid \lambda)$ is in the same family as the prior $q(\beta \mid \eta)$. Given these conditions, the batch update for $q(\beta \mid \lambda)$ in Eq. 7 is

$$\lambda = \eta + \sum_{i=1}^{n} \mathbb{E}_{\hat{q}(z_i)}\left[\bar{t}(x_i, z_i)\right]. \tag{10}$$

The term $\bar{t}(x_i, z_i)$ is defined as $\bar{t}(x_i, z_i) \triangleq [t(x_i, z_i); 1]$.

Analysis in [12, 13, 29] shows that given the conditional conjugacy assumption, the batch update of parameter $\lambda$ in Eq. 10 can be easily turned into a stochastic update using natural gradient [38]. Suppose our parameter is $\lambda_t$ at step $t$. Given a random observation $x_t$, we sample from $q(z_t \,|\, x_t, \lambda_t)$ to obtain the empirical distribution $\hat{q}(z_t)$. With an appropriate learning rate $\rho_t$, we have

$$\lambda_{t+1} \leftarrow \lambda_t + \rho_t \left( -\lambda_t + \eta + n \mathbb{E}_{\hat{q}(z_i)} \left[ \bar{t}(x_t, z_t) \right] \right). \tag{11}$$

This corresponds to an stochastic update using the noisy natural gradient to optimize the lower bound in Eq. 3 [39]. (We note that the natural gradient is an approximation since our $q(z_i)$ in Eq. 6 is suboptimal for the lower bound Eq. 3.)

*Mini-batch.* A common strategy used in stochastic variational inference [12, 13] is to use a small batch of samples at each update. Suppose we have a batch size $S$, and the set of samples $x_t, t \in \mathbb{S}$. Using our formulation, the $q(z_t, t \in \mathbb{S})$ becomes

$$q(z_{t,t \in \mathbb{S}}) \propto \mathbb{E}_{q(\beta \,|\, \lambda_t)} \left[ \prod_{t \in \mathbb{S}} p(x_t, z_t | \beta) \right].$$

We choose not to factorize $z_{t,t \in \mathbb{S}}$, since factorization will potentially lead to the label-switching problem when new components are instantiated for BNP models [7].

## 2.3  Truncation-free stochastic variational inference for BNP models

We have described locally collapsed variational inference in a general setting. Our main interests in this paper are BNP models, and we now show how this approach leads to truncation-free variational algorithms. We describe the approach for a DP mixture model [21], whose full description was presented in the beginning of §2.1. See S.2 for the details on the HDP topic models [3].

**The global variational distribution.**  The variational distribution for the global hidden variables, mixture components $\beta$ and stick proportions $\bar{\pi}$ is

$$q(\theta, \bar{\pi} \,|\, \lambda, u, v) = \prod_k q(\theta \,|\, \lambda_k) q(\bar{\pi}_k \,|\, u_k, v_k),$$

where $\lambda_k$ is the Dirichlet parameter and $(u_k, v_k)$ is the Beta parameter. The sufficient statistic term $t(x_i, z_i)$ defined in Eq. 9 can be summarized as

$$t(x_i, z_i)_{\lambda_{kw}} = 1_{[z_i=k]} \sum_j 1_{[x_{ij}=w]}; \quad t(x_i, z_i)_{u_k} = 1_{[z_i=k]}, \quad t(x_i, z_i)_{v_k} = \sum_{j=k+1} 1_{[z_i=j]},$$

where $1_{[\cdot]}$ is the indicator function. Suppose at time $t$, we have obtained the empirical distribution $\hat{q}(z_i)$ for observation $x_i$, we use Eq. 11 to update Dirichlet parameter $\lambda$ and Beta parameter $(u, v)$,

$$\lambda_{kw} \leftarrow \lambda_{kw} + \rho_t(-\lambda_{kw} + \eta + n\hat{q}(z_i = k) \sum_j 1_{[x_{ij}=w]})$$
$$u_k \leftarrow u_k + \rho_t(-u_k + 1 + n\hat{q}(z_i = k))$$
$$v_k \leftarrow v_k + \rho_t(-v_k + a + n \sum_{\ell=k+1} \hat{q}(z_i = \ell)).$$

Although we have a unbounded number of mixture components, we do not need to represent them explicitly. Suppose we have $T$ components that are associated with some data. These updates indicate $q(\theta_k \,|\, \lambda_k) = \text{Dirichlet}(\eta)$ and $q(\bar{\pi}_k) = \text{Beta}(1, a)$, i.e., their prior distributions, when $k > T$. Similar to a Gibbs sampler [6], the model is "truncated" automatically. (We re-ordered the sticks according to their sizes [15].)

**The local empirical distribution $\hat{q}(z_i)$.**  Since the mixture assignment $z_i$ is the only hidden variable, we obtain its analytical form using Eq. 6,

$$q(z_i = k) \propto \int p(x_i \,|\, \theta_k) p(z_i = k \,|\, \pi) q(\theta_k \,|\, \lambda_k) q(\bar{\pi}) \mathrm{d}\theta_k \mathrm{d}\bar{\pi}$$
$$= \frac{\Gamma(\sum_w \lambda_{kw})}{\prod_w \Gamma(\lambda_{kw})} \frac{\prod_w \Gamma(\lambda_{kw} + \sum_j 1_{[x_{ij}=w]})}{\Gamma(\sum_w \lambda_{kw} + |x_i|)} \frac{u_k}{u_k + v_k} \prod_{\ell=1}^{k-1} \frac{v_\ell}{u_\ell + v_\ell},$$

where $|x_i|$ is the document length and $\Gamma(\cdot)$ is the Gamma function. (For mini batches, we do not have an analytical form, but we can sample from it.) The probability of creating a new component is

$$q(z_i > T) \propto \frac{\Gamma(\eta V)}{\Gamma^V(\eta)} \frac{\prod_w \Gamma(\eta + \sum_j 1_{[x_{ij}=w]})}{\Gamma(\eta V + |x_i|)} \prod_{k=1}^T \frac{v_k}{u_k + v_k}.$$

We sample from $q(z_i)$ to obtain its empirical distribution $\hat{q}(z_i)$. If $z_i > T$, we create a new component.

**Discussion.** Why is "locally collapsed" enough? This is analogous to the collapsed Gibbs sampling algorithm in DP mixture models [6]— whether or not exploring a new mixture component is initialized by *one single sample*. The locally collapsed variational inference is powerful enough to trigger this. In the toy example above, the role of distribution $q(\theta_2) = \mathrm{Dirichlet}(0.1, \ldots, 0.1)$ is similar to that of the potential new component we want to maintain in Gibbs sampling for DP mixture models.

Note the difference between this approach and those found in [17, 18], which use mean-field methods that can grow or shrink the truncation using split-merge moves. These approaches are model-specific and difficult to design. Further, they do not transfer to the stochastic setting. In contrast, the approach presented here grows the truncation as a natural consequence of the inference algorithm and is easily adapted to stochastic inference.

# 3  Experiments

We evaluate our methods on DP mixtures and HDP topic models, comparing them to truncation-based stochastic mean-field variational inference. We focus on stochastic methods and large data sets.

**Datasets.** We analyzed two large document collections. The *Nature* data contains about 350K documents from the journal *Nature* from years 1869 to 2008, with about 58M tokens and a vocabulary size of 4,253. The *New York Times* dataset contains about 1.8M documents from the years 1987 to 2007, with about 461M tokens and a vocabulary size of 8,000. Standard stop words and those words that appear less than 10 times or in more than 20 percent of the documents are removed, and the final vocabulary is chosen by TF-IDF. We set aside a test set of 10K documents from each corpus on which to evaluate its predictive power; these test sets were not given for training.

**Evaluation Metric.** We evaluate the different algorithms using held-out per-word likelihood,

$$\mathrm{likelihood}_{\mathrm{pw}} \triangleq \log p(\mathcal{D}_{\mathrm{test}} \,|\, \mathcal{D}_{\mathrm{train}})/\textstyle\sum_{x_i \in \mathcal{D}_{\mathrm{test}}} |x_i|,$$

Higher likelihood is better. Since exact computing the held-out likelihood is intractable, we use approximations. See S.3 for details of approximating the likelihood. There is some question as to the meaningfulness of held-out likelihood as a metric for comparing different models [40]. Held-out likelihood metrics are nonetheless suited to measuring how well an inference algorithm accomplishes the specific optimization task defined by a model.

**Experimental Settings.** For DP mixtures, we set component Dirichlet parameter $\eta = 0.5$ and the concentration parameter of DP $a = 1$. For HDP topic models, we set topic Dirichlet parameter $\eta = 0.01$, and the first-level and second-level concentration parameters of DP $a = b = 1$ as in [13]. (See S.2 for the full description of HDP topic models.) For stochastic mean-field variational inference, we set the truncation level at 300 for both DP and HDP. We run all algorithms for 10 hours and took the model at the final stage as the output, without assessing the convergence. We vary the mini-batch size $S = \{1, 2, 5, 10, 50, 100, 500\}$. (We do not intend to compare DP and HDP; we want to show our algorithm works on both methods.)

For stochastic mean-field approach, we set the learning rate according to [13] with $\rho_t = (\tau_0 + t)^{-\kappa}$ with $\kappa = 0.6$ and $\tau_0 = 1$. We start our new method with 0 components without seeing any data. We cannot use the learning rate schedule as in [13], since it gives very large weights to the first several components, effectively leaving no room for creating new components on the fly. We set the learning rate $\rho_t = S/n_t$, for $n_t < n$, where $n_t$ is the size of corpus that the algorithm has seen at time $t$. After we see all the documents, $n_t = n$. For both stochastic mean-field and our algorithm, we set the lower bound of learning rate as $S/n$. We found this works well in practice. This mimics the usual trick of running Gibbs sampler—one uses sequential prediction for initialization and after all data points have been initialized, one runs the full Gibbs sampler [41]. We remove components with fewer than 1 document for DP and topics with fewer than 1 word for HDP topic models each time when we process 20K documents.

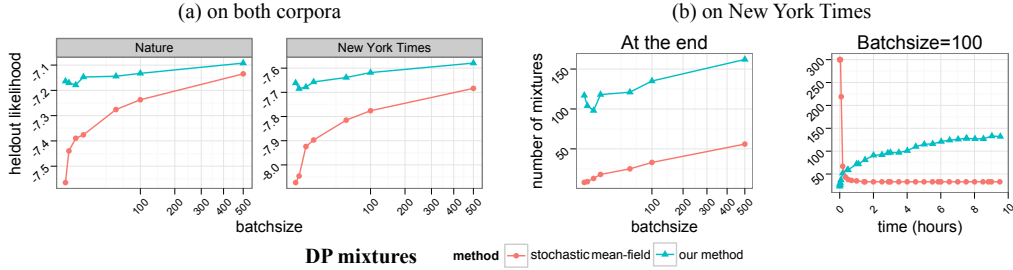

Figure 3: Results on DP mixtures. (a) Held-out likelihood comparison on both corpora. Our approach is more robust to batch sizes and gives better predictive performance. (b) The inferred number of mixtures on New York Times. (Nature is similar.) The left of figure (b) shows the number of mixture components inferred after 10 hours; our method tends to give more mixtures. Small batch sizes for the stochastic mean-field approach do not really work, resulting in very small number of mixtures. The right of figure (b) shows how different methods infer the number of mixtures. The stochastic mean field approach shrinks it while our approach grows it.

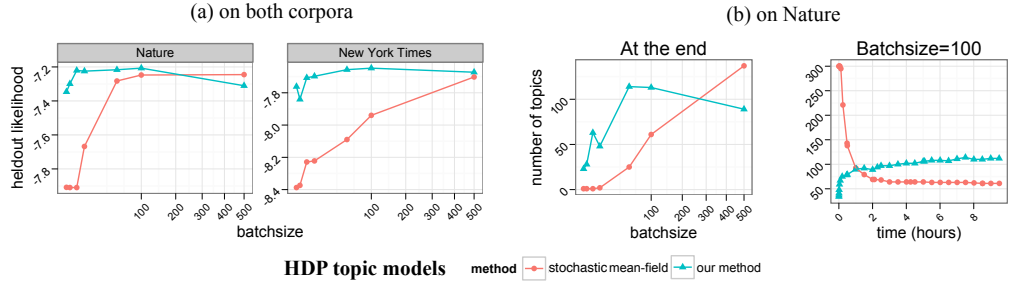

Figure 4: Results on HDP topic models. (a) Held-out likelihood comparison on both corpora. Our approach is more robust to batch sizes and gives better predictive performance most of time. (b) The inferred number of topics on Nature. (New York Times is similar.) The left of figure (b) show the number of topics inferred after 10 hours; our method tends to give more topics. Small batch sizes for the stochastic mean-field approach do not really work, resulting in very small number of topics. The right of figure (b) shows how different methods infer the number of topics. Similar to DP, the stochastic mean field approach shrinks it while our approach grows it.

**Results.** Figure 3 shows the results for DP mixture models. Figure 3(a) shows the held-out likelihood comparison on both datasets. Our approach is more robust to batch sizes and usually gives better predictive performance. Small batch sizes of the stochastic mean-field approach do not work well. Figure 3(b) shows the inferred number of mixtures on New York Times. (Nature is similar.) Our method tends to give more mixtures than the stochastic mean-field approach. The stochastic mean-field approach shrinks the preset truncation; our approach does not need a truncation and grows the number of mixtures when data requires.

Figure 4 shows the results for HDP topic models. Figure 4(a) shows the held-out likelihood comparison on both datasets. Similar to DP mixtures, our approach is more robust to batch sizes and gives better predictive performance most of time. And small batch sizes of the stochastic mean-field approach do not work well. Figure 4(b) shows the inferred number of topics on Nature. (New York Times is similar.) This is also similar to DP. Our method tends to give more topics than the stochastic mean-field approach. The stochastic mean-field approach shrinks the preset truncation while our approach grows the number of topics when data requires.

One possible explanation that our method gives better results than the truncation-based stochastic mean-field approach is as follows. For truncation-based approach, the algorithm relies more on the random initialization placed on the parameters within the preset truncations. If the random initialization is not used well, performance degrades. This also explains that smaller batch sizes in stochastic mean-fields tend to work much worse—the first fewer samples might dominate the effect from the random initialization, leaving no room for later samples. Our approach mitigates this problem by allowing new components/topics to be created as data requires.

If we compare DP and HDP, the best result of DP is better than that of HDP. But this comparison is not meaningful. Besides the different settings of hyperparameters, computing the held-out likelihood for DP is tractable, but intractable for HDP. We used importance sampling to approximate. (See S.3

for details.) [42] shows that importance sampling usually gives the correct ranking of different topic models but significantly underestimates the probability.

# 4 Conclusion and future work

In this paper, we have developed truncation-free stochastic variational inference algorithms for Bayesian nonparametric models (BNP models) and applied them to two large datasets. Extensions to other BNP models, such as Pitman-Yor process [43], Indian buffet process (IBP) [23, 24] and the nested Chinese restaurant process [18, 25] are straightforward by using their stick-breaking constructions. Exploring how this algorithm behaves in the true streaming setting where the program never stops—a "never-ending learning machine" [44]—is an interesting future direction.

**Acknowledgements.** Chong Wang was supported by Google PhD and Siebel Scholar Fellowships. David M. Blei is supported by ONR N00014-11-1-0651, NSF CAREER 0745520, AFOSR FA9550-09-1-0668, the Alfred P. Sloan foundation, and a grant from Google.

# References

[1] Hjort, N., C. Holmes, P. Mueller, et al. *Bayesian Nonparametrics: Principles and Practice*. Cambridge University Press, Cambridge, UK, 2010.

[2] Antoniak, C. Mixtures of Dirichlet processes with applications to Bayesian nonparametric problems. *The Annals of Statistics*, 2(6):1152–1174, 1974.

[3] Teh, Y., M. Jordan, M. Beal, et al. Hierarchical Dirichlet processes. *Journal of the American Statistical Association*, 101(476):1566–1581, 2007.

[4] Andrieu, C., N. de Freitas, A. Doucet, et al. An introduction to MCMC for machine learning. *Machine Learning*, 50:5–43, 2003.

[5] Jordan, M., Z. Ghahramani, T. Jaakkola, et al. Introduction to variational methods for graphical models. *Machine Learning*, 37:183–233, 1999.

[6] Neal, R. Markov chain sampling methods for Dirichlet process mixture models. *Journal of Computational and Graphical Statistics*, 9(2):249–265, 2000.

[7] Newman, D., A. Asuncion, P. Smyth, et al. Distributed algorithms for topic models. *Journal of Machine Learning Research*, 10:1801–1828, 2009.

[8] Smola, A., S. Narayanamurthy. An architecture for parallel topic models. *Proc. VLDB Endow.*, 3(1-2):703–710, 2010.

[9] Ahmed, A., M. Aly, J. Gonzalez, et al. Scalable inference in latent variable models. In *International Conference on Web Search and Data Mining (WSDM)*. 2012.

[10] Wainwright, M., M. Jordan. Graphical models, exponential families, and variational inference. *Foundations and Trends in Machine Learning*, 1(1–2):1–305, 2008.

[11] Hoffman, M., D. M. Blei, C. Wang, et al. Stochastic Variational Inference. *ArXiv e-prints*, 2012.

[12] Hoffman, M., D. Blei, F. Bach. Online inference for latent Drichlet allocation. In *Advances in Neural Information Processing Systems (NIPS)*. 2010.

[13] Wang, C., J. Paisley, D. M. Blei. Online variational inference for the hierarchical Dirichlet process. In *International Conference on Artificial Intelligence and Statistics (AISTATS)*. 2011.

[14] Blei, D., M. Jordan. Variational inference for Dirichlet process mixtures. *Journal of Bayesian Analysis*, 1(1):121–144, 2005.

[15] Kurihara, K., M. Welling, Y. Teh. Collapsed variational Dirichlet process mixture models. In *International Joint Conferences on Artificial Intelligence (IJCAI)*. 2007.

[16] Teh, Y., K. Kurihara, M. Welling. Collapsed variational inference for HDP. In *Advances in Neural Information Processing Systems (NIPS)*. 2007.

[17] Kurihara, K., M. Welling, N. Vlassis. Accelerated variational Dirichlet process mixtures. In *Advances in Neural Information Processing Systems (NIPS)*. 2007.

[18] Wang, C., D. Blei. Variational inference for the nested Chinese restaurant process. In *Advances in Neural Information Processing Systems (NIPS)*. 2009.

[19] Gelman, A., J. Hill. *Data Analysis Using Regression and Multilevel/Hierarchical Models*. Cambridge Univ. Press, 2007.

[20] McLachlan, G., D. Peel. *Finite mixture models*. Wiley-Interscience, 2000.

[21] Escobar, M., M. West. Bayesian density estimation and inference using mixtures. *Journal of the American Statistical Association*, 90:577–588, 1995.

[22] Blei, D., A. Ng, M. Jordan. Latent Dirichlet allocation. *Journal of Machine Learning Research*, 3:993–1022, 2003.

[23] Griffiths, T., Z. Ghahramani. Infinite latent feature models and the Indian buffet process. In *Advances in Neural Information Processing Systems (NIPS)*. 2006.

[24] Teh, Y., D. Gorur, Z. Ghahramani. Stick-breaking construction for the Indian buffet process. In *International Conference on Artifical Intelligence and Statistics (AISTATS)*. 2007.

[25] Blei, D., T. Griffiths, M. Jordan. The nested Chinese restaurant process and Bayesian nonparametric inference of topic hierarchies. *Journal of the ACM*, 57(2):1–30, 2010.

[26] Sethuraman, J. A constructive definition of Dirichlet priors. *Statistica Sinica*, 4:639–650, 1994.

[27] Bishop, C. *Pattern Recognition and Machine Learning*. Springer New York., 2006.

[28] Zhai, K., J. Boyd-Graber, N. Asadi, et al. Mr. LDA: A flexible large scale topic modeling package using variational inference in MapReduce. In *International World Wide Web Conference (WWW)*. 2012.

[29] Sato, M. Online model selection based on the variational Bayes. *Neural Computation*, 13(7):1649–1681, 2001.

[30] Opper, M., O. Winther. *From Naive Mean Field Theory to the TAP Equations*, pages 1–19. MIT Press, 2001.

[31] MacKay, D. *Information Theory, Inference, and Learning Algorithms*. Cambridge University Press, 2003.

[32] Asuncion, A., M. Welling, P. Smyth, et al. On smoothing and inference for topic models. In *Uncertainty in Artificial Intelligence (UAI)*. 2009.

[33] Sato, I., H. Nakagawa. Rethinking collapsed variational Bayes inference for LDA. In *International Conference on Machine Learning (ICML)*. 2012.

[34] Sato, I., K. Kurihara, H. Nakagawa. Practical collapsed variational bayes inference for hierarchical dirichlet process. In *International Conference on Knowledge Discovery and Data Mining*, KDD, pages 105–113. ACM, New York, NY, USA, 2012.

[35] Minka, T. Divergence measures and message passing. Tech. Rep. TR-2005-173, Microsoft Research, 2005.

[36] Teh, Y., D. Newman, M. Welling. A collapsed variational Bayesian inference algorithm for latent Dirichlet allocation. In *Advances in Neural Information Processing Systems (NIPS)*. 2006.

[37] Mimno, D., M. Hoffman, D. Blei. Sparse stochastic inference for latent dirichlet allocation. In *International Conference on Machine Learning (ICML)*. 2012.

[38] Amari, S. Natural gradient works efficiently in learning. *Neural computation*, 10(2):251–276, 1998.

[39] Robbins, H., S. Monro. A stochastic approximation method. *The Annals of Mathematical Statistics*, 22(3):pp. 400–407, 1951.

[40] Chang, J., J. Boyd-Graber, C. Wang, et al. Reading tea leaves: How humans interpret topic models. In *Advances in Neural Information Processing Systems (NIPS)*. 2009.

[41] Griffiths, T., M. Steyvers. Finding scientific topics. *Proceedings of the National Academy of Sciences (PNAS)*, 2004.

[42] Wallach, H., I. Murray, R. Salakhutdinov, et al. Evaluation methods for topic models. In *International Conference on Machine Learning (ICML)*. 2009.

[43] Pitman, J., M. Yor. The two-parameter poisson-dirichlet distribution derived from a stable subordinator. *The Annals of Probability*, 25(2):855–900, 1997.

[44] Carlson, A., J. Betteridge, B. Kisiel, et al. Toward an architecture for never-ending language learning. In *AAAI Conference on Artificial Intelligence (AAAI)*. 2010.

